# Transductive Inference for Estimating Values of Functions

**Olivier Chapelle[*], Vladimir Vapnik[*,†], Jason Weston[††,†,*]**
[*] AT&T Research Laboratories, Red Bank, USA.
[†] Royal Holloway, University of London, Egham, Surrey, UK.
[††] Barnhill BioInformatics.com, Savannah, Georgia, USA.
*{chapelle,vlad,weston}@research.att.com*

## Abstract

We introduce an algorithm for estimating the values of a function at a set of test points $x_{\ell+1}, \ldots, x_{\ell+m}$ given a set of training points $(x_1, y_1), \ldots, (x_\ell, y_\ell)$ without estimating (as an intermediate step) the regression function. We demonstrate that this direct (transductive) way for estimating values of the regression (or classification in pattern recognition) can be more accurate than the traditional one based on two steps, first estimating the function and then calculating the values of this function at the points of interest.

## 1  Introduction

Following [6] we consider a general scheme of transductive inference. Suppose there exists a function $y^* = f_0(x)$ from which we observe the measurements corrupted with noise

$$((x_1, y_1), \ldots (x_\ell, y_\ell)), \quad y_i = y_i^* + \xi_i. \tag{1}$$

Find an algorithm $A$ that using both the given set of training data (1) and the given set of test data

$$(x_{\ell+1}, \ldots, x_{\ell+m}) \tag{2}$$

selects from a set of functions $\{x \mapsto f(x)\}$ a function

$$y = f(x) = f_A(x|x_1, y_1, \ldots, x_\ell, y_\ell, x_{\ell+1}, \ldots, x_{\ell+m}) \tag{3}$$

and minimizes at the points of interest the functional

$$R(A) = \mathrm{E}\left( \sum_{i=\ell+1}^{\ell+m} (y_i^* - f_A(x_i|x_1, y_1, \ldots, x_\ell, y_\ell, x_{\ell+1}, \ldots, x_{\ell+m}))^2 \right) \tag{4}$$

where expectation is taken over $x$ and $\xi$. For the training data we are given the vector $x$ and the value $y$, for the test data we are only given $x$.

Usually, the problem of estimating values of a function at points of interest is solved in two steps: first in a given set of functions $f(x, \alpha)$, $\alpha \in \Lambda$ one estimates the regression, i.e the function which minimizes the functional

$$R(\alpha) = \int ((y - f(x, \alpha))^2 dF(x, y), \tag{5}$$

(the inductive step) and then using the estimated function $y = f(x, \alpha_\ell)$ we calculate
the values at points of interest

$$y_i^* = f(x_i^*, \alpha_\ell) \tag{6}$$

(the deductive step).

Note, however, that the estimation of a function is equivalent to estimating its values in the continuum points of the domain of the function. Therefore, by solving the regression problem using a restricted amount of information, we are looking for a more general solution than is required. In [6] it is shown that using a direct estimation method one can obtain better bounds than through the two step procedure.

In this article we develop the idea introduced in [5] for estimating the values of a function only at the given points.

The material is organized as follows. In Section 1 we consider the classical (inductive) Ridge Regression procedure, and the leave–one–out technique which is used to measure the quality of its solutions. Section 2 introduces the transductive method of inference for estimation of the values of a function based on this leave–one–out technique. In Section 3 experiments which demonstrate the improvement given by transductive inference compared to inductive inference (in both regression and pattern recognition) are presented. Finally, Section 4 summarizes the results.

## 2    Ridge Regression and the Leave–One–Out procedure

In order to describe our transductive method, let us first discuss the classical two-step (inductive plus deductive) procedure of Ridge Regression. Consider the set of functions linear in their parameters

$$f(x, \alpha) = \sum_{i=1}^{n} \alpha_i \phi_i(x). \tag{7}$$

To minimize the expected loss (5), where $F(x, y)$ is unknown, we minimize the following empirical functional (the so–called Ridge Regression functional [1])

$$R_{emp}(\alpha) = \frac{1}{\ell} \sum_{i=1}^{\ell} (y_i - f(x_i, \alpha))^2 + \gamma ||\alpha||^2 \tag{8}$$

where $\gamma$ is a fixed positive constant, called the regularization parameter. The minimum is given by the vector of coefficients

$$\alpha_\ell = \alpha(x_1, y_1, \ldots, x_\ell, y_\ell) = (K^T K + \gamma I)^{-1} K^T Y \tag{9}$$

where

$$Y = (y_1, \ldots, y_\ell)^T, \tag{10}$$

and $K$ is a matrix with elements:

$$K_{ij} = \phi_j(x_i), \quad i = 1, \ldots, \ell, \quad j = 1, \ldots, n. \tag{11}$$

The problem is to choose the value $\gamma$ which provides small expected loss for training on a sample $S_\ell = \{(x_1, y_1), \ldots, (x_\ell, y_\ell)\}$.

For this purpose, we would like to choose $\gamma$ such that $f_\gamma$ minimizing (8) also minimizes

$$R = \int (y^* - f_\gamma(x^*|S_\ell))^2 dF(x^*, y^*) dF(S_\ell). \tag{12}$$

Since $F(x, y)$ is unknown one cannot estimate this minimum directly. To solve this problem we instead use the leave–one–out procedure, which is an almost unbiased estimator of (12). The leave–one–out error of an algorithm on the training sample $S_\ell$ is

$$T_{loo}(\gamma) = \frac{1}{\ell} \sum_{i=1}^{\ell} \left(y_i - f_\gamma(x_i | S_\ell \setminus (x_i, y_i))\right)^2 . \tag{13}$$

The leave–one–out procedure consists of removing from the training data one element (say $(x_i, y_i)$), constructing the regression function only on the basis of the remaining training data and then testing the removed element. In this fashion one tests all $\ell$ elements of the training data using $\ell$ different decision rules. The minimum over $\gamma$ of (13) we consider as the minimum over $\gamma$ of (12) since the expectation of (13) coincides with (12) [2].

For Ridge Regression, one can derive a closed form expression for the leave–one–out error. Denoting

$$A_\gamma^{-1} = (K^T K + \gamma I)^{-1} \tag{14}$$

the error incurred by the leave–one–out procedure is [6]

$$T_{loo}(\gamma) = \frac{1}{\ell} \sum_{i=1}^{\ell} \left( \frac{y_i - k_i^T A_\gamma^{-1} K^T Y}{1 - k_i^T A_\gamma^{-1} k_i} \right)^2 \tag{15}$$

where

$$k_t = (\phi_1(x_t) \ldots, \phi_n(x_t))^T . \tag{16}$$

Let $\gamma = \gamma^0$ be the minimum of (15). Then the vector

$$Y^0 = K^*(K^T K + \gamma^0 I)^{-1} K^T Y \tag{17}$$

where

$$K^* = \begin{pmatrix} \phi(x_{\ell+1}) & \cdots & \phi_n(x_{\ell+1}) \\ \vdots & & \vdots \\ \phi_1(x_{\ell+m}) & \cdots & \phi_n(x_{\ell+m}) \end{pmatrix} \tag{18}$$

is the Ridge Regression estimate of the unknown values $(y_{\ell+1}^*, \ldots, y_{\ell+m}^*)$.

## 3  Leave–One–Out Error for Transductive Inference

In transductive inference, our goal is to find an algorithm $A$ which minimizes the functional (4) using both the training data (1) and the test data (2). We suggest the following method: predict $(y_{\ell+1}^*, \ldots, y_{\ell+m}^*)$ by finding those values which minimize the leave–one–out error of Ridge Regression training on the joint set

$$(x_1, y_1), \ldots, (x_\ell, y_\ell), (x_{\ell+1}, y_{\ell+1}^*), \ldots, (x_{\ell+m}, y_{\ell+m}^*). \tag{19}$$

This is achieved in the following way. Suppose we treat the unknown values $(y_{\ell+1}^*, \ldots, y_{\ell+m}^*)$ as variables and for some fixed value of these variables we minimize the following empirical functional

$$R_{emp}(\alpha | y_1^*, \ldots, y_m^*) = \frac{1}{\ell + m} \left( \sum_{i=1}^{\ell} (y_i - f(x_i, \alpha))^2 + \sum_{i=\ell+1}^{\ell+m} (y_i^* - f(x_i, \alpha))^2 \right) + \gamma \|\alpha\|^2. \tag{20}$$

This functional differs only in the second term from the functional (8) and corresponds to performing Ridge Regression with the extra pairs

$$(x_{\ell+1}, y_{\ell+1}^*), \ldots, (x_{\ell+m}, y_{\ell+m}^*). \tag{21}$$

Suppose that vector $Y^* = (y_1^*, \ldots, y_m^*)$ is taken from some set $Y^* \in \mathcal{Y}$ such that the pairs (21) can be considered as a sample drawn from the same distribution as the pairs $(x_1, y_1^*), \ldots, (x_\ell, y_\ell^*)$. In this case the leave–one–out error of minimizing (20) over the set (19) approximates the functional (4). We can measure this leave–one–out error using the same technique as in Ridge Regression. Using the closed form (15) one obtains

$$T_{loo}(\gamma | y_1^*, \ldots, y_m^*) = \frac{1}{\ell + m} \sum_{i=1}^{\ell+m} \left( \frac{\hat{Y}_i - \hat{k}_i^T \hat{A}_\gamma^{-1} \hat{K}^T \hat{Y}}{1 - \hat{k}_i^T \hat{A}_\gamma^{-1} \hat{k}_i} \right)^2. \tag{22}$$

where we denote $\hat{x} = (x_1, \ldots, x_{\ell+m})$, $\hat{Y} = (y_1, \ldots, y_\ell, y_{\ell+1}^*, \ldots, y_{\ell+m}^*)^T$, and

$$\hat{A}_\gamma^{-1} = (\hat{K}^T \hat{K} + \gamma I)^{-1} \tag{23}$$

$$\hat{K}_{ij} = \phi_j(\hat{x}_i), \quad i = 1, \ldots, \ell+m, \quad j = 1, \ldots, n. \tag{24}$$

$$\hat{k}_t = (\phi_1(\hat{x}_t) \ldots, \phi_n(\hat{x}_t))^T. \tag{25}$$

Now let us rewrite the expression (22) in an equivalent form to separate the terms with $\hat{Y}$ from the terms with $x$. Introducing

$$C = I - \hat{K} A_\gamma^{-1} \hat{K}^T, \tag{26}$$

and the matrix $M$ with elements

$$M_{ij} = \sum_{k=1}^{\ell+m} \frac{C_{ik} C_{kj}}{C_{kk}^2} \tag{27}$$

we obtain the equivalent expression of (22)

$$T_{loo}(\gamma | y_1^*, \ldots, y_m^*) = \frac{1}{\ell + m} (\hat{Y}^T M \hat{Y}). \tag{28}$$

In order for the $Y^*$ which minimize the leave–one–out procedure to be valid it is required that the pairs (21) are drawn from the same distribution as the pairs $(x_1, y_1^*), \ldots, (x_\ell, y_\ell^*)$. To satisfy this constraint we choose vectors $Y^*$ from the set

$$\mathcal{Y} = \{Y^* : \|Y^* - Y^0\| \leq R\} \tag{29}$$

where the vector $Y^0$ is the solution obtained from classical Ridge Regression.

To minimize (28) under constraint (29) we use the functional

$$T_{loo}^{\gamma^*}(\gamma) = \hat{Y}^T M \hat{Y} + \gamma^* \|Y^* - Y^0\|^2 \tag{30}$$

where $\gamma^*$ is a constant depending on $R$.

Now, to find the values at the given points of interest (2) all that remains is to find the minimum of (30) in $Y^*$. Note that the matrix $M$ is obtained using only the vectors $\hat{x}$. Therefore, to find the minimum of this functional we rewrite Equation (30) as

$$T_{loo}^{\gamma^*}(\gamma) = Y^T M_0 Y + 2 Y^{*T} M_1 Y + Y^{*T} M_2 Y^* + \gamma^* \|Y^* - Y^0\|^2 \tag{31}$$

where

$$M = \begin{vmatrix} M_0 & M_1 \\ M_1^T & M_2 \end{vmatrix} \tag{32}$$

and $M_0$ is a $\ell \times \ell$ matrix, $M_1$ is a $\ell \times m$ matrix and $M_2$ is a $m \times m$ matrix. Taking the derivative of (31) in $Y^*$ we obtain the condition for the solution

$$2M_1Y + 2M_2Y^* - 2\gamma^*Y^0 + 2\gamma^*Y^* = 0 \qquad (33)$$

which gives the predictions

$$Y^* = (\gamma^*I + M_2)^{-1}(-M_1Y + \gamma^*Y^0). \qquad (34)$$

In this algorithm (which we will call Transductive Regression) we have two parameters to control: $\gamma$ and $\gamma^*$. The choice of $\gamma$ can be found using the leave-one-out estimator (15) for Ridge Regression. This leaves $\gamma^*$ as the only free parameter.

## 4 Experiments

To compare our one–step transductive approach with the classical two–step approach we performed a series of experiments on regression problems. We also describe experiments applying our technique to the problem of pattern recognition.

### 4.1 Regression

We conducted computer simulations for the regression problem using two datasets from the DELVE repository: **boston** and **kin-32fh**.

The **boston** dataset is a well–known problem where one is required to estimate house prices according to various statistics based on 13 locational, economic and structural features from data collected by the U.S Census Service in the Boston Massachusetts area.

The **kin-32fh** dataset is a realistic simulation of the forward dynamics of an 8 link all-revolute robot arm. The task is to predict the distance of the end-effector from a target, given 32 inputs which contain information on the joint positions, twist angles and so forth.

Both problems are nonlinear and contain noisy data. Our objective is to compare our transductive inference method directly with the inductive method of Ridge Regression. To do this we chose the set of basis functions $\phi_i(x) = \exp\left(-||x - x_i||^2/2\sigma^2\right)$, $i = 1, \ldots, \ell$, and found the values of $\gamma$ and $\sigma$ for Ridge Regression which minimized the leave–one–out bound (15). We then used the same values of these parameters in our transductive approach, and using the basis functions $\phi_i(x) = \exp\left(-||x - \hat{x}_i||^2/2\sigma^2\right)$, $i = 1, \ldots, \ell + m$, we then chose a fixed value of $\gamma^*$.

For the **boston** dataset we followed the same experimental setup as in [4], that is, we partitioned the training set of 506 observations randomly 100 times into a training set of 481 observations and a testing set of 25 observations. We chose the values of $\gamma$ and $\sigma$ by taking the minimum average leave–one–out error over five more random splits of the data stepping over the parameter space. The minimum was found at $\gamma = 0.005$ and $\log \sigma = 0.7$. For our transductive method, we also chose the parameter $\gamma^* = 10$. In Figure 1a we plot mean squared error (MSE) on the test set averaged over the 100 runs against $\log \sigma$ for Ridge Regression and Transductive Regression. Transductive Regression outperforms Ridge Regression, especially at the minimum.

To observe the influence of the number of test points $m$ on the generalization ability of our transductive method, we ran further experiments, setting $\gamma^* = \ell/2m$ for

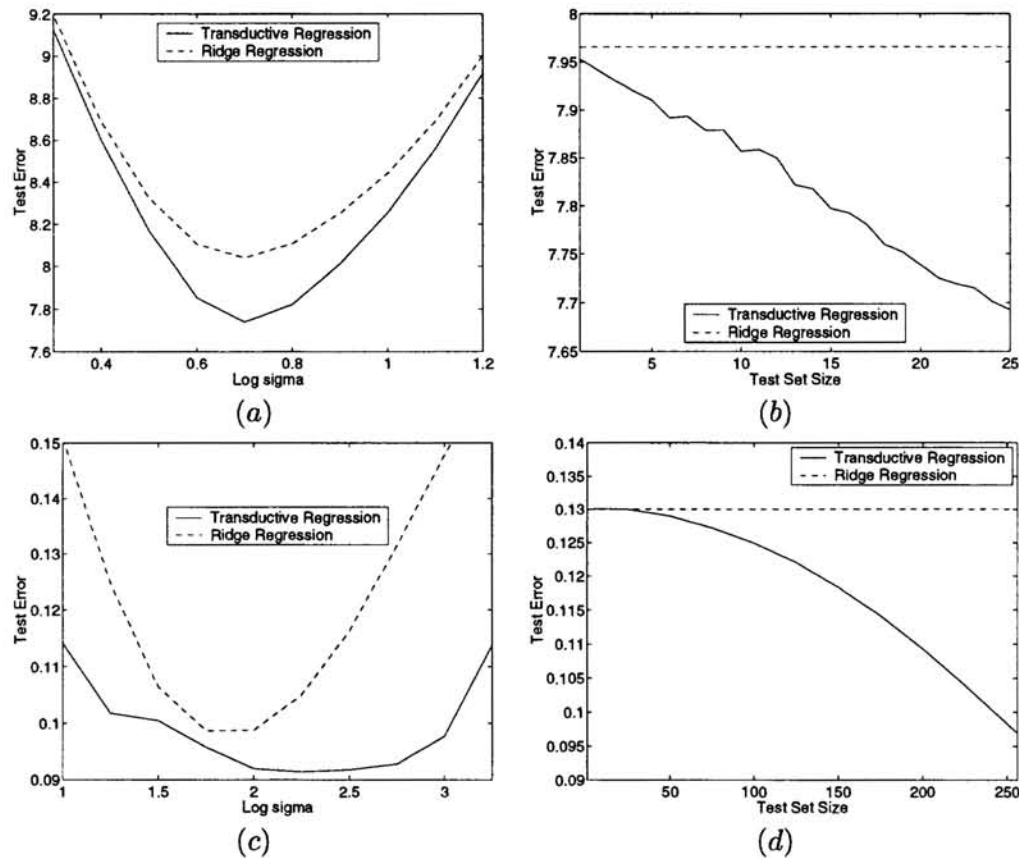

Figure 1: *A comparison of Transductive Regression to Ridge Regression on the* **boston** *dataset: (a) error rates for varying* $\sigma$, *(b) varying the test set size,* $m$, *and on the* **kin-32fh** *dataset: (c) error rates for varying* $\sigma$, *(d) varying the test set size.*

different values of $m$. In Figure 1b we plot $m$ against MSE on the testing set, at $\log \sigma = 0.7$. The results indicate that increasing the test set size gives improved performance in Transductive Regression. For Ridge Regression, of course, the size of the testing set has no influence on the generalization ability.

We then performed similar experiments on the **kin-32fh** dataset. This time, as we were interested in large testing sets giving improved performance for Transductive Regression we chose 100 splits where we took a subset of only 64 observations for training and 256 for testing. Again the leave–one–out estimator was used to find the values $\gamma = 0.1$ and $\log \sigma = 2$ for Ridge Regression, and for Transductive Regression we also chose the parameter $\gamma^* = 0.1$. We plotted MSE on the testing set against $\log \sigma$ (Figure 1c) and the size of the test set $m$ for $\log \sigma = 2.75$ (also, $\gamma^* = 50/m$) (Figure 1d) for the two algorithms. For large test set sizes our method outperforms Ridge Regression.

## 4.2   Pattern Recognition

This technique can also be applied for pattern recognition problems by solving them based on minimizing functional (8) with $y = \pm 1$. Such a technique is known as a Linear Discriminant (LD) technique.

|  | AB | AB$_R$ | SVM | TLD |
|---|---|---|---|---|
| Postal | – | – | 5.5 | 4.7 |
| Banana | 12.3 | 10.9 | 11.5 | 11.4 |
| Diabetes | 26.5 | 23.8 | 23.5 | 23.3 |
| Titanic | 22.6 | 22.6 | 22.4 | 22.4 |
| Breast Cancer | 30.4 | 26.5 | 26.0 | 25.7 |
| Heart | 20.3 | 16.6 | 16.0 | 15.7 |
| Thyroid | 4.4 | 4.6 | 4.8 | 4.0 |

Table 1: *Comparison of percentage test error of AdaBoost (AB), Regularized AdaBoost (AB$_R$), Support Vector Machines (SVM) and Transductive Linear Discrimination (TLD) on seven datasets.*

Table 1 describes results of experiments on classification in the following problems: 2 class digit recognition ($0 - 4$ versus $5 - 9$) splitting the training set into 23 runs of 317 observations and considering a testing set of 2000 observations, and six problems from the UCI database. We followed the same experimental setup as in [3]: the performance of a classifier is measured by its average error over one hundred partitions of the datasets into training and testing sets. Free parameter(s) are chosen via validation on the first five training datasets. The performance of the transductive LD technique was compared to Support Vector Machines, AdaBoost and Regularized AdaBoost [3].

It is interesting to note that in spite of the fact that LD technique is one of the simplest pattern recognition techniques, transductive inference based upon this method performs well compared to state of the art methods of pattern recognition.

## 5   Summary

In this article we performed transductive inference in the problem of estimating values of functions at the points of interest. We demonstrate that estimating the unknown values via a one–step (transductive) procedure can be more accurate than the traditional two–step (inductive plus deductive) one.

## References

[1] A. Hoerl and R. W. Kennard. Ridge regression: Biased estimation for nonorthogonal problems. *Technometrics*, 12(1):55–67, 1970.

[2] A. Luntz and V. Brailovsky. On the estimation of characters obtained in statistical procedure of recognition,. *Technicheskaya Kibernetica*, 1969. [In Russian].

[3] G. Rätsch, T. Onoda, and K.-R. Müller. Soft margins for adaboost. Technical report, Royal Holloway, University of London, 1998. TR–98–21.

[4] C. Saunders, A. Gammermann, and V. Vovk. Ridge regression learning algorithm in dual variables. In *Proccedings of the 15th International Conference on Machine Learning*, pages 515–521. Morgan Kaufmann, 1998.

[5] V. Vapnik. Estimating of values of regression at the point of interest. In *Method of Pattern Recognition*. Sovetskoe Radio, 1977. [In Russian].

[6] V. Vapnik. *Estimation of Dependences Based on Empirical Data*. Springer–Verlag, New York, 1982.